# Whence Sparseness?

**C. van Vreeswijk**
Gatsby Computational Neuroscience Unit
University College London
17 Queen Square, London WC1N 3AR, United Kingdom

## Abstract

It has been shown that the receptive fields of simple cells in V1 can be explained by assuming optimal encoding, provided that an extra constraint of sparseness is added. This finding suggests that there is a reason, independent of optimal representation, for sparseness. However this work used an ad hoc model for the noise. Here I show that, if a biologically more plausible noise model, describing neurons as Poisson processes, is used sparseness does not have to be added as a constraint. Thus I conclude that sparseness is not a feature that evolution has striven for, but is simply the result of the evolutionary pressure towards an optimal representation.

## 1 Introduction

Recently there has been an resurgence of interest in using optimal coding strategies to 'explain' the response properties of neuron in the primary sensory areas [1]. Notably this approach was used Olshausen and Field [2] to infer the receptive field of simple cells in the primary visual cortex. To arrive at the correct results however, they had to add sparseness of activity as an extra constraint. Others have shown that similar results are obtained if one assumes that the neurons represent independent components of natural stimuli [3]. The fact that these studies need to impose an extra constraint suggests strongly that the subsequent processing of the stimuli uses either sparseness or independence of the neuronal activity. It is therefore highly important to determine whether these constraints are really necessary. Here it will be argued that the necessity of the sparseness constraint in these models is due to modeling the noise in the system incorrectly. Modeling the noise in a biologically more plausibly way leads to a representation of the input in which the sparseness of the activity naturally follows from the optimality of the representation.

## 2 Gaussian Noise

Several approaches have been used to find an output that represents the input optimally, for example, minimizing the square difference between the input and its reconstruction. In this paper I will concentrate on a different definition of optimality, I require that the mutual

information between the input and output is maximized. If the number of output units is at least equal to the dimensionality of the input space a perfect reconstruction of the input is possible, unless there is noise in the system. So for an (over)-complete representation optimal encoding only makes sense in the presence of noise. It is important to note that the optimal solution depends on the model of noise that is taken, even if one takes the limit where the noise goes to zero. Thus it is important to have an adequate noise model.

Most optimal coding schemes describe the neuronal output by an input-dependent mean to which Gaussian noise is added. This is, roughly speaking, also the implicit assumption in an optimization procedure in which the mean square reconstruction error is minimized, but it is also often used explicitly when the the mutual information is maximized. It is instructive to see, in the latter case, why one needs to impose extra constraints to obtain un-ambiguous results: Assume the input $s$ has dimension $N_i$ and is drawn from a distribution $p(s)$. There are $N_o \geq N_i$ output neurons whose rates $r$ satisfy

$$r = Ws + \sigma\xi, \tag{1}$$

where $\xi$ is a $N_o$ dimensional univariate Gaussian with zero mean, $p_\xi(\xi) = (2\pi)^{-N_o/2} \exp(-\xi^T\xi/2)$ (the superscript $T$ denotes the transpose). The task is to find the $N_o \times N_i$ matrix $W_m$ that maximizes the mutual information $I_M$ between $r$ and $s$, defined by [4]

$$I_M(r, s) = \int dr \int ds\, p(r, s)\{\log[p(r|s)] - \log[\int ds'p(r, s')]\}. \tag{2}$$

Here $p(r, s)$ is the joint probability distribution of $r$ and $s$ and $p(r|s)$ is the conditional probability of $r$, given $s$. It is immediately clear that replacing $W$ by $cW$ with $c > 1$ increases the mutual information by effectively reducing the noise by a factor $1/c$. Thus maximal mutual information is obtained as the rates become infinite. Thus, to get sensible results, a constraint has to be placed on the rates. A natural constraint in this framework is a constraint on the average square rates $r$, $<r^Tr> = N_oR_0^2$. Here I have used $<\cdots>$ to denote the average over the noise and inputs and $R_0^2 > \sigma^2$ is the mean square rate.

Under this constraint, however, the optimal solution is still vastly degenerate. Namely if $W_M$ is a matrix that gives the maximum mutual information, for any unitary matrix $U$ ($U^TU = 1$), $UW_m$ will also maximize $I_M$. This is straightforward to show. For $r = W_ms + \sigma\xi$ the mutual information is given by

$$\begin{aligned} I_M(r, s; W_m) &= \int dr \int ds\, p(s)p_\xi\left(\frac{r - W_ms}{\sigma}\right)\left\{\log\left[p_\xi\left(\frac{r - W_ms}{\sigma}\right)\right] - \right. \\ &\quad \left. \log\left[\int ds'\, p(s')p_\xi\left(\frac{r - W_ms'}{\sigma}\right)\right]\right\}, \end{aligned} \tag{3}$$

where I have used $I_M(r, s; W)$ to denote the mutual information when the matrix $W$ is used. In the case where $r$ satisfies $r = UW_ms + \sigma\xi$ the mutual information is given by equation 3, with $W_m$ replaced by $UW_m$. Changing variables from $r$ to $r' = U^Tr$ and using $|\det(U)| = 1$, this can be rewritten as

$$\begin{aligned} I_M(r, s; UW_m) &= \int dr' \int ds\, p(s)p_\xi\left(U\frac{r' - W_ms}{\sigma}\right)\left\{\log\left[p_\xi\left(U\frac{r' - W_ms}{\sigma}\right)\right] - \right. \\ &\quad \left. \log\left[\int ds'p(s')p_\xi\left(U\frac{r' - W_ms'}{\sigma}\right)\right]\right\}. \end{aligned} \tag{4}$$

Because $p_\xi(\xi)$ is a function of $\xi^T \xi$ only, $p_\xi(U\xi) = p_\xi(\xi)$, and therefore $I_m(r, s; UW_m) = I_m(r, s, W_m)$. In other words, because we have assumed a model in which the noise is described by independent Gaussians, or generally the distribution of the noise $\xi$ is a function of $\xi^T \xi$ only, the mutual information is invariant to unitary transformations of the output. Clearly, this degeneracy is a result of the particular choice of the noise statistics and unlikely to survive when we try to account for biologically observed noise more accurately. In the latter case it may well happen that the degeneracy is broken in such a way that maximizing the mutual information with a constraint on the average rates is itself sufficient to obtain a sparse representation.

## 3 Poisson Noise

To obtain a robust insight in this issue, it is important that the system can be treated analytically. The desire for biologically plausibility of the system should therefore be balanced by the desire to keep it simple. Ubiquitous features found in electrophysiological experiments likely to be of importance are (see for example [5]): i) Neurons transmit information through spikes. ii) Consecutive inter-spike intervals are at best weakly correlated. iii) With repeated presentation of the stimulus the variance in the number of spikes a neuron emits over a given period varies nearly linearly with the mean number of emitted spikes.

A simple model that captures all these features of the biological system is the Poisson process [6]. I will thus consider a system in which the neurons are described by such a process. The general model is as follows: The inputs are given by an $N_i$ dimensional vector $s$ drawn from a distribution $p(s)$. These give rise to $N_o$ inputs $u$ into the cells, which satisfy $u = Ws$, where $W$ is the coupling matrix. The inputs $u$ are transformed into rates through a transfer function $g$, $r_i = g(u_i)$. The output of the network is observed for a time $T$. Optimal encoding of the input is defined by the maximal mutual information between the spikes the neurons emit and the stimulus. Let $n_i$ be the total number of spikes for neuron $i$ and $n$ the $N_o$ dimensional array of spike counts, then $p(n|r) = \prod_i (r_i T)^{n_i} \exp(-r_i T)/n_i!$. Optimal coding is achieved by determining $W_m$ such that

$$W_m = \text{argmax}_W (I_M(n, s; W)). \tag{5}$$

As before there is need for a constraint on $W$ so that solutions with infinite rates are excluded. Whereas with Gaussian channels fixing the mean square rates is the most natural choice for the constraint, for Poissonian neurons it is more natural to fix the mean number of emitted spikes, $\sum_i < r_i >= N_o R_0$. By rescaling time we can, without loss of generality, assume that $R_0 = 1$.

## 4 A Simple Example

The simplest example in which we can consider whether such systems will lead a sparse representation is a system with a single neuron and a 1 dimensional input, which is uniformly distributed between 0 and 1. Assume that the unit has output rate $r = 0$ when the input satisfies $s < 1 - p$ and rate $1/p$ if $s > 1 - p$. Because the neuron is either 'on' or 'off', maximal information about its state can be obtained by checking whether it fired either one or more spikes or did not fire at all in the time-window over which the neuron was observed. If the neuron is 'on', the probability that it does not spike in a time $T$ is $1 - \exp(-T/p)$, otherwise it is 1. Thus the probability distribution is

$$p(0, s) = 1 - e^{-T/p}\Theta(s - 1 + p), \quad p(1+, s) = e^{-T/p}i\Theta(s - 1 + p), \tag{6}$$

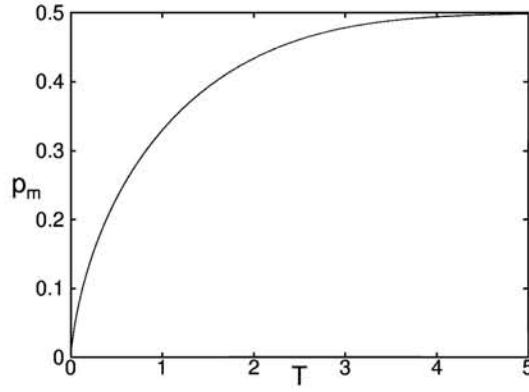

Figure 1: $p_m$, the value of $p$ that maximizes the mutual information as function of the measuring time-window $T$.

where I have used $p(1+, s)$ to denote the probability of 1 or more spikes and an input $s$. The mutual information satisfies

$$
\begin{aligned}
I_M(n, s; p) = \ & p(1 - e^{-T/p})\log(1 - e^{-T/p}) - pe^{-T/p}\log(p) - \\
& (1 - p(1 - e^{-T/p}))\log(1 - p(1 - e^{-T/p})). \quad (7)
\end{aligned}
$$

Figure 1 shows $p_m$, the value of $p$ that maximizes the mutual information, as a function of the time $T$ over which the neuron is observed. For small $T$, $p_m$ is small, this reflects the fact that the reliability of the neuron is increased if the rate in the 'on' state $(1/p)$ is made maximal. For large $T$, $p_m$ approaches $1/2$, the value for which the entropy of the output rate $r$ is maximized. We thus see a trade-off between the reliability which wants to make $p$ as small as possible, and the capacity, which pushes $p$ to $1/2$. For time intervals that are smaller than or on the order of the mean 1: inter-spike interval the former dominates and leads to an optimal solution in which the neuron is, with a high probability, quiet, or, with a low probability, fires vigorously. Thus in this system the neurons fire sparsely if the measuring time is sufficiently short.

## 5  A More Interesting Example

Somewhat closer to the problem of optimal encoding in V1, but still tractable, is the following example. A two-dimensional input $s$ is drawn from a distribution $p(s)$ given by

$$
p(s_1, s_2) = \frac{1}{2}\left(\delta(s_1)e^{-|s_2|/2} + e^{-|s_1|/2}\delta(s_2)\right). \quad (8)
$$

This input is encoded by four neurons, the inputs into these neurons are given by

$$
\begin{pmatrix} u_1 \\ u_2 \end{pmatrix} = \frac{1}{|\cos(\phi)| + |\sin(\phi)|}\begin{pmatrix} \cos(\phi) & \sin(\phi) \\ -\sin(\phi) & \cos(\phi) \end{pmatrix}\begin{pmatrix} s_1 \\ s_2 \end{pmatrix}, \quad (9)
$$

$u_3 = -u_1$, and $u_4 = -u_2$. The rates $r_i$ satisfy $r_i = (u_i)_+ \equiv (u_i + |u_i|)/2$, the threshold linear function. Due to the symmetry of the problem, rotation by a multiple of $\pi/2$ leads to the same rates, up to a permutation. Thus we can restrict ourselves to $0 \le \phi < \pi/2$.

It is straightforward to show that $\sum_i < n_i >= 4$, and that sparseness of the activity, here defined by $\sum_i (< n_i^2 > - < n_i >^2)/ < n_i >^2$, has its minimum for $\phi = \pi/4$, and

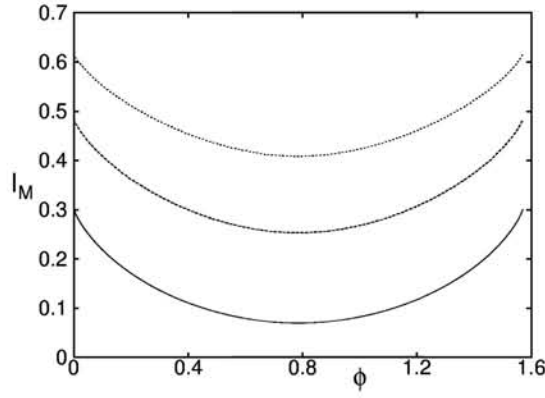

Figure 2: Mutual information $I_M$ as function of $\phi$, for $T = 1$ (solid line), for $T = 2$ (dashed line), and $T = 3$ (dotted line).

maximum for $\phi = 0$. Some straightforward algebra shows that the mutual information is given by

$$
\begin{aligned}
I_M(\boldsymbol{n}, \boldsymbol{s}; \phi) = {} & \gamma T + \log(1 + T) - \sum_{n=0}^{\infty} \left( \frac{T}{1+T} \right)^n \left[ \log \left( \frac{n}{1+T} \right) + \right. \\
& \frac{1}{1+T} \left( \frac{|\cos(\phi)|}{|\cos(\phi)| + |\sin(\phi)|} \right)^n \log \left( 1 + \left| \frac{\sin(\phi)}{\cos(\phi)} \right|^n \right) + \\
& \left. \frac{1}{1+T} \left( \frac{|\sin(\phi)|}{|\cos(\phi)| + |\sin(\phi)|} \right)^n \log \left( 1 + \left| \frac{\cos(\phi)}{\sin(\phi)} \right|^n \right) \right]
\end{aligned}
\tag{10}
$$

Figure 2 shows the $I_M$ as a function of $\phi$ for different values of $T$. For all values of $T$ the mutual information is maximal when $\phi = 0$ and minimal for $\pi/4$. For large $T$ the angular dependent part of $I_M$ scales as $1/T$. So this dependence becomes negligible if the output is observed for a long time. Yet as in the previous example, for relatively short time-windows, optimal coding automatically leads to sparse coding.

## 6   Optimal Coding in V1

Finally we turn to optimal encoding of images in the striate cortex. To study this I consider a system in with a large number, $K$, of natural images. The intensity of pixel $j$ ($1 \le j \le N_i$) of image $\kappa$ is given by $s_j(\kappa)$. These images induce an input $u_i$ into neuron $i$ ($1 \le i \le N_o$) given by

$$
u_i(\kappa) = \sum_j W_{ij} s_j(\kappa),
\tag{11}
$$

which lead to firing rates $r_i(\kappa)$ which satisfy $r_i(\kappa) = \beta^{-1} \log(1 + e^{\beta u_i(\kappa)})$. Here I have used as smoothened version of of the threshold linear function (which is recovered in the limit $\beta \to \infty$) to ensure that its derivative with respect to $W_{ij}$ is continuous. The neurons fire in a Poissonian manner, so that for image $\kappa$ the probability $P(\boldsymbol{n}, \kappa)$ of observing $n_i$ spikes for neuron $i$ is given by

$$
p(\boldsymbol{n}, \kappa) = \prod_{i=1}^{N_o} \frac{(r_i(\kappa)T)^{n_i}}{n_i!} e^{-r_i(\kappa)T}.
\tag{12}
$$

We want to choose the matrix such $W$ the mutual information $I_M$ between the image $\kappa$ and the spike count for the different neurons, given by

$$I_M(\boldsymbol{n}, \kappa; W) = \sum_{\boldsymbol{n}} \frac{1}{K} \sum_{\kappa} p(\boldsymbol{n}, \kappa; W) \left[ \log(p(\boldsymbol{n}, \kappa; W)) - \log(p(\boldsymbol{n}; W)) \right] \qquad (13)$$

is maximized. Obviously an analytic solution is out of the question, but one may want to try to approach the optimal solution by gradient assent, using

$$\frac{\partial I_M(\boldsymbol{n}, \kappa; W)}{\partial W_{ij}} = \sum_{\boldsymbol{n}} \frac{1}{K} \sum_{\kappa} \frac{\partial p(\boldsymbol{n}, \kappa; W)}{\partial W_{i,j}} \left[ \log(p(\boldsymbol{n}, \kappa; W)) - \log(p(\boldsymbol{n}; W)) \right] \qquad (14)$$

for the derivative of the mutual information. Here the derivative of $p(\boldsymbol{n}, \kappa)$ is given by $\partial p(\boldsymbol{n}, \kappa)/\partial W_{i,j} = s_j(\kappa)(1 - e^{\beta r_i(\kappa)}(n_i/r_i - T)p(\boldsymbol{n}, \kappa)$. The constraint on the rates, $K^{-1} \sum_{\kappa} \sum_i r_i(\kappa) = N_o$ is incorporated by adding this function with a Lagrange multiplier to the objective function.

Unfortunately the gradient assent approach is impractical, since the summation over $\boldsymbol{n}$ scales exponentially with $N_o$. In any case, one may want to use stochastic gradient assent to avoid getting trapped in local minima. But to do stochastic gradient assent it is sufficient to obtain a unbiased estimate of $\partial I_M/\partial W_{ij}$. Denoting this derivative by $\partial I_M/\partial W_{ij} = \sum_{\boldsymbol{n}} F_{ij}(\boldsymbol{n})$, where $F_{ij}$ has the obvious meaning, one can rewrite the derivative of the mutual information as

$$\frac{\partial I_M}{\partial W_{ij}} = \sum_{\boldsymbol{n}} \tilde{p}(\boldsymbol{n}) \frac{F_{ij}(\boldsymbol{n})}{\tilde{p}(\boldsymbol{n})} \qquad (15)$$

provided that $\tilde{p}(\boldsymbol{n})$ is non-zero for every $\boldsymbol{n}$ for which $p(\boldsymbol{n}; W) \neq 0$. An unbiased estimate of $\partial I_M/\partial W_{ij}$ (denoted by $\partial \tilde{I}_M/\partial W_{ij}$) is obtained by taking

$$\frac{\partial \tilde{I}_M}{\partial W_{ij}} = \frac{1}{L} \sum_{\ell=1}^{L} \frac{F_{ij}(\boldsymbol{n}(\ell))}{\tilde{p}(\boldsymbol{n}(\ell))}, \qquad (16)$$

where the $L$ vectors $\boldsymbol{n}(\ell)$ are drawn independently from the distribution $\tilde{p}(\boldsymbol{n})$. Conjecturing that $F_{ij}(\boldsymbol{n})$ is roughly proportional to $p(\boldsymbol{n}; W)$, I set $\tilde{p}(\boldsymbol{n}) = p(\boldsymbol{n}; W)$ to obtain the best estimate of $\partial I_M/\partial W_{ij}$ for fixed $L$. Drawing from $p(\boldsymbol{n}; W)$ can be done in a computationally cheap way by first randomly picking $\kappa$ and then draw from $p(\boldsymbol{n}, \kappa, W)$, which factorizes.

In the simulation of the system I have used the natural image collection from [7]. From each of the approximately 4000 images a $10 \times 10$ patch. These were preprocessed by subtracting the mean and whitening the image. To reduce the effects of the corners a circular region was then extracted from the images. This resulted in an input which has 80 pixel intensities per image. These pixel intensities were encoded by 160 neurons. The coupling matrix was initialized by drawing tits components independently from a Gaussian distribution and rescaling the matrix to normalize $< r_i >$. The time-window $T$ was chosen to be $T = 0.5$, and $\beta$ was gradually increased from its initial value of $\beta = 1$ to $\beta = 10$. The coupling matrix was updated using $\epsilon = 10^{-4}$, and $L = 10$.

Figure 3 shows the some of the receptive fields that were obtained after the system had approached the fixed point, e.i. the running average of the mutual information no longer increased. These receptive fields look rather similar to those obtained from simple cells in the striate cortex. However a more thorough analysis of these receptive field and the sparseness of the rate distribution still has to be undertaken.

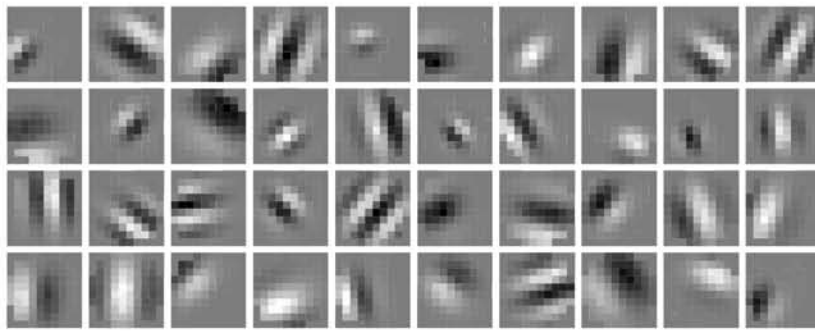

Figure 3: Forty examples of receptive fields that show clear Gabor like structure.

## 7  Discussion

I have shown why optimal encoding using Gaussian channels naturally leads to highly degenerate solutions necessitating extra constraints. Using the biologically more plausible noise model which describes the neurons by Poisson processes naturally leads to a sparse representation, when optimal encoding is enforced, for two analytically tractable models. For a model of the striate cortex Poisson statistics also leads to a network in which the receptive fields of the neurons mimic those of V1 simple cells, without the need to *impose* sparseness. This leads to the conclusion that sparseness is not an independent constraint that is imposed by evolutionary pressure, but rather is a consequence of optimal encoding.

## References

[1] Baddeley, R., Hancock, P., and Földiák, P. (eds.) (2000) *Information Theory and the Brain.* (Cambridge University Press, Cambridge).

[2] Olshausen, B.A. and Field, D.J. (1996) **Nature** 381:607; (1998) **Vision Research** 37:3311.

[3] Bell, A.j. and Sejnowski, T.J. (1997) **Vision Res.** 37:3327; van Hateren, J.H. and van der Schaaf, A. (1998) **Proc. R. Soc. Lond.** 265:359.

[4] Cover, T.M. and Thomas, J.A. (1991) *Information Theory* (Whiley and Sons, New York).

[5] Richmond, B.J., Optican, L.M., and Spitzer, H. (1990) **J. Neurophysiol.** 64:351; Rolls, E.T., Critchley, H.D., and Treves, A. (1996) **J. Neurophysiol.** 75:1982; Dean, A.F. (1981) **Exp. Brain. Res.** 44:437.

[6] Smith, W.L. (1951) **Biometrica** 46:1.

[7] van Hateren, J.H. and van der Schaaf, A. (1998) Proc.R.Soc.Lond. B 265:359-366.
